# Rational inference of relative preferences

**Nisheeth Srivastava**
Dept of Computer Science
University of Minnesota

**Paul R Schrater**
Dept of Psychology
University of Minnesota

## Abstract

Statistical decision theory axiomatically assumes that the relative desirability of different options that humans perceive is well described by assigning them option-specific scalar utility functions. However, this assumption is refuted by observed human behavior, including studies wherein preferences have been shown to change systematically simply through variation in the set of choice options presented. In this paper, we show that interpreting desirability as a relative comparison between available options at any particular decision instance results in a rational theory of value-inference that explains heretofore intractable violations of rational choice behavior in human subjects. Complementarily, we also characterize the conditions under which a rational agent selecting optimal options indicated by dynamic value inference in our framework will behave identically to one whose preferences are encoded using a static ordinal utility function.

## 1 Introduction

Normative theories of human choice behavior have long been based on how economic theory has postulated they should be made. The standard version of the theory states that consumers seek to maximize innate, stable preferences over the options they consume. Preferences are represented by numerical encoding of value in terms of utilities, and subjects are presumed to select the option with the maximum expected utility. The most difficult part of this theory is that preferences must exist before decisions can be made. The standard response, in both economics and decision theory, to the basic question *"Where do preferences come from?"* is *"We'll leave that one to the philosophers, utilities are simply abstractions we assume for the work we do."*, which, while true, is not an answer.

While this question has been studied before in the form of learning utility values from behavior [5, 14, 10], human preferences exhibit patterns of behavior that are impossible to reconcile with the idea that stable numerical representations of value can be ascribed to each item they choose between. Behavioral experiments in the last half century have conclusively demonstrated (see [18] for a comprehensive review) that human choice strongly violates the key axioms that the existence of stable utility values depends on. A particular subset of these violations, called context effects, wound the utility maximization program the most deeply, since such violations cannot be explained away as systematic distortions of underlying utility and/or probability representations [22]. Consider for instance, the "frog legs" thought problem, pictured in Figure 1, introduced by Luce and Raiffa in their seminal work [15]. No possible algebraic reformulation of option-specific utility functions can possibly explain preference reversals of the type exhibited in the frog legs example. Preference reversals elicited through choice set variation have been observed in multiple empirical studies, using a variety of experimental tasks, and comprise one of the most powerful criticisms of the use of expected utility as a normative standard in various economic programs, e.g. in public goods theory. However, for all its problems, the mathematical simplicity of the utility framework and lack of principled alternatives has allowed it to retain its central role in microeconomics [12], machine learning [1], computational cognitive science [7] and neuroscience [11].

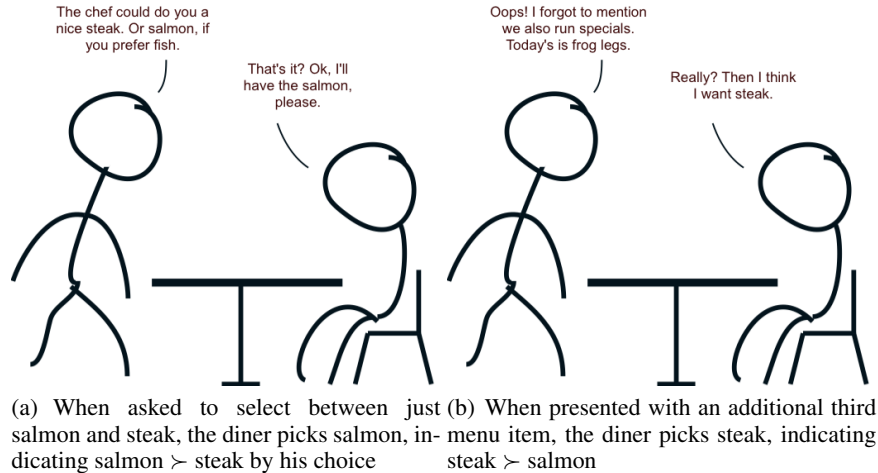

(a) When asked to select between just salmon and steak, the diner picks salmon, indicating salmon $\succ$ steak by his choice

(b) When presented with an additional third menu item, the diner picks steak, indicating steak $\succ$ salmon

Figure 1: Illustration of Luce's 'frog legs' thought experiment. No possible absolute utility assignation to individual items can account for the choice behavior exhibited by the diner in this experiment. The frog legs example is illustrative of reversals in preference occuring solely through variation in the set of options a subject has to choose from.

Our contribution in this paper is the development of a rational model that infers preferences from limited information about the *relative* value of options. We postulate that there is a value inference process that predicts the relative goodness of items in enabling the agent to achieve its homeostatic and other longer-range needs (e.g. survival and reproductive needs). While this process should be fully explicated, we simply don't know enough to make detailed mathematical models. However, we show that we only have to postulate that feedback from decisions provides limited information about the relative worth of options within the choice set for a decision to retrieve an inductive representation of value that is equivalent to traditional preference relations. Thus, instead of assuming utilities as being present in the environment, we learn an equivalent sense of option desirability from information in a limited format that depends on the set of options in the decision set. This inductive methodology naturally makes choice sets informative about the value of options, and hence affords simple explanations for context effects. We show how to formalize the idea of relative value inference, and that it provides a new *rational* foundation for understanding the origins of human preferences.

## 2 Human Preferences via Value Inference

We begin by reviewing and formalizing probabilistic decision-making under uncertainty. An agent selects between possibilities $x$ in the world represented by the set $\mathcal{X}$. The decision-making problem can be formulated as one wherein the agent forms a belief $b(x), x \in \mathcal{X}$ about the relative **desirability** of different possibilities in $\mathcal{X}$ and uses this belief to choose an element or subset $\mathcal{X}^* \subset \mathcal{X}$. When these beliefs satisfy the axioms of utility, the belief function simply the expected utility associated with individual possibilities $u(x), u : \mathcal{X} \to \mathbb{R}$.

We assume these desirabilities must be learned from experience, suggesting a reinforcement learning approach. The agent's belief about the relative desirability of the world is constantly updated by information that it receives about the desirability of options in terms of value signals $r(x)$. Belief updating produces transition dynamics on $b^t(x)$. Given a sequence of choices, the normative expectation is for agents to select possibilities in a way that maximizes their infinite-horizon cumulative discounted desirability,

$$\arg\max_{x^{(t)}} \sum_t^{\infty} \gamma^t b^t(x). \tag{1}$$

The sequence of choices selected describes the agent's expected desirability maximizing behavior in a belief MDP-world.

From a Bayesian standpoint, it is critical to describe the belief updating about the desirability of different states. Let $p(x|r^{(1:t)})$ represent the belief a value $x$ is the best option given a sequence of value signals. Since the agent learns this distribution from observing $r(x)$ signals from the environment, an update of the form,

$$p(x|r^{(t)}) = p(r^{(t)}|x) \times p(x|\{r^{(1)}, r^{(2)} \cdots, r^{(t-1)}\}), \tag{2}$$

reflects the basic process of belief formation via value signals. When value signals are available for every option, independent of other options, the likelihood term $p(r|x)$ in Equation (2) is a probabilistic representation of observed utility, which remains unaffected in the update by the agent's history of sampling past possibilities and hence is invariant to transition probabilities. Such separation between utilities and probabilities in statistical decision theory is called *probabilistic sophistication*, an axiom that underlies almost all existing computational decision theory models [11].

The crux of our new approach is that we assume that value signals $p(r|x)$ are **not available for every option**. Instead, we assume we get partial information about the value of one or more options within the set of options $c$ available in the decision instance $t$. In this case value signals are hidden for most options $x$. However, the set of options $c \in \mathcal{C} \subseteq \mathcal{P}(\mathcal{X})^1$ observed can now potentially be used as auxiliary information to impute values for options whose value has not been observed. In such a scenario, the agent requires a more sophisticated inference process,

$$
\begin{aligned}
p(x|r^{(1:t)}) &= \frac{1}{p(r^{(1:t)})} \int_c p(x, c, r^{(1:t)}), \\
&= \frac{1}{p(r^{(1:t)})} \int_c [p(r^t|x, c)p(c|x)] \times p(x|\{r^{(1)}, r^{(2)} \cdots, r^{(t-1)}\}).
\end{aligned}
$$

Importantly, we concentrate on understanding the meaning of *utility* in this framework. As in the case of value observability for all options, a probabilistic representation of utility under indirect observability must be equivalent to,

$$p(r|x) = \frac{p(r, x)}{p(x)} = \frac{\int_c p(r, x, c)}{\int_c p(x|c)p(c)} = \frac{\int_c p(r|x, c)p(x|c)p(c)}{\int_c p(x|c)p(c)}. \tag{3}$$

The resulting prediction of value of an option couples value signals received across decision instances with different option sets, or contexts. The intuition behind this approach is contained in the frog leg's example - the set of options become informative about the hidden state of the world, like whether the restaurant has a good chef.

Naively, one could assume that altering existing theory to include this additional source of information would be an incremental exercise. However, a formidable epistemological difficulty arises as soon as we attempt to incorporate context into utility-based accounts of decision-making. To see this, let us assume that we have defined a measure of utility $u(x, c)$ that is sensitive to the context $c$ of observing possibility $x$. Now, for such a utility measure, if it is true that for any two possibilities $\{x_i, x_j\}$ and any two contexts $\{c_k, c_l\}$,

$$u(x_i, c_k) > u(x_j, c_k) \Rightarrow u(x_i, c_l) > u(x_j, c_l),$$

then the choice behavior of an agent maximizing $u(x, c)$ would be equivalent to one maximizing $u(x)$. Thus, for the inclusion of context to have any effect, there must exist at least some $\{x_i, x_j, c_k, c_l\}$ for which the propositions $u(x_i, c_k) > u(x_j, c_k)$ and $u(x_i, c_l) < u(x_j, c_l)$ can hold simultaneously.

Note however, that the context in this operationalization is simply a collection of other possibilities, i.e. $c \subseteq \mathcal{X}$ which ultimately implies $u(x, c) = u(\mathcal{X}^*) = u(\mathcal{X}), \mathcal{X}^* = \{x, c\} \subseteq \mathcal{X}$. Such a measure could assign absolute numbers to each of the possibilities, but any such static assignment would make it impossible for the propositions $u(x_1, \mathcal{X}) > u(x_2, \mathcal{X})$ and $u(x_1, \mathcal{X}) < u(x_2, \mathcal{X})$ to hold simultaneously, as is desired of a context-sensitive utility measure. Thus, we see that it is impossible to design a utility function $u$ such that $u : \mathcal{X} \times \mathcal{C} \to \mathbb{R}$. If we wish to incorporate the effects of context variation on the desirability of a particular world possibility, we must abandon a foundational premise of existing statistical decision theory - the representational validity of absolute utility.

---

$^1\mathcal{P}(\cdot)$ references the power set operation throughout this paper.

# 3 Rational decisions without utilities

In place of the traditional utility framework, we define an alternative conceptual partitioning of the world $\mathcal{X}$ as a discrete choice problem. In this new formulation, at any decision instant $t$, agents observe the feasibility of a subset $o^{(t)} \subseteq \mathcal{X}$ of all the possibilities in the world. In the following exposition, we use $y^t$ to denote an indicator function on $\mathcal{X}$ encoding the possibilities observed as $o^{(t)}$,

$$y^t(x) = \sum_{i \in o^{(t)}} \delta(x - i),$$

. An intelligent agent will encode its understanding of partial observability as a belief over which possibilities of the world likely co-occur. We call an agent's belief about the co-occurrence of possibilities in the world its understanding of the *context* of its observation. We instantiate contexts $c$ as subsets of $\mathcal{X}$ that the agent *believes* will co-occur based on its history of partial observations of the world and index them with an indicator function $z$ on $\mathcal{X}$, so that for context $c^{(t)}$,

$$z^t(x) = \sum_{i \in c^{(t)}} \delta(x - i).$$

Instead of computing absolute utilities on all $x \in \mathcal{X}$, a context-aware agent evaluates the comparable **desirability** of only those possibilities considered feasible in a particular context $c$. Hence, instead of using scalar values to indicate which possibility is more preferable, we introduce preference information into our system via a desirability function $d$ that simply 'points' to the best option in a given context, i.e. $d^{(c)} = B$, where $B$ is a binary relation $(c, c, m)$ and $m_i = 1$ iff $c_i \succ c_{i'} \forall c_{i'} \in c \setminus \{c_i\}$ and zero otherwise. The desirability indicated by $d^{(c)}$ can be remapped on to the larger set of options by defining a relative desirability across all possibilities $r(x) = m, x \in c$ and zero otherwise.

Recall now that we have already defined what we mean by utility in our system in Equation 3. Instantiated in the discrete choice setting, this can be restated as a probabilistic definition of relative desirability at decision instant $t$ as,

$$R^{(t)}(x) = p(r^{(t)}|x) = \frac{\sum_c^{\mathcal{C}} p(r^{(t)}|x, c)p(x|c)p(c)}{\sum_c^{\mathcal{C}} p(x|c)p(c)}, \tag{4}$$

where it is understood that $p(c) = p(c|\{o_1, o_2, \cdots, o_{t-1}\})$ is a distribution on the set of all possible contexts inferred from the agent's observation history. From the definition of desirability, we can also obtain a simple definition of $p(r|x, c)$ as $p(r_i|x_i, c) = 1$ iff $r_i x_i = 1$ and zero otherwise. To instantiate Eqn (4) concretely, it is finally necessary to define a specific form for the likelihood term $p(x|c)$. While multiple mathematical forms can be proposed for this expression, depending on quantitative assumptions about the amount of uncertainty intrinsic to the observation, the underlying intuition must remain one that obtains the highest possible value for $c = o$ and penalizes mismatches in set membership. Such definitions can be introduced in the mathematical definition of the element-wise mismatch probability, $p(\neg y_i^t|z_i^t)$. Since $p(x_i|c^{(t)}) = 1 - p(\neg y_i^t|z_i^t)$, we can use these element-wise probabilities to compute the likelihood of any particular observation $o^{(t)}$ as,

$$P(o^{(t)}|c^{(t)}) = 1 - p\left(\bigcup_i^{|o^{(t)}|} \{\neg y_i^t\} | \bigcup_i^{|c^{(t)}|} \{z_i^t\}\right), = 1 - \beta \sum_i^{|\mathcal{X}|} p(\neg y_i^t|z_i^t),$$

where $\beta$ is a parameter controlling the magnitude of the penalty imposed for each mismatch observed.

This likelihood function can then be used to update the agent's posterior belief about the contexts it considers viable at decision instance $t$, given its observation history as,

$$p(c^{(t)}|\{o^{(1)}, o^{(2)}, \cdots, o^{(t)}\}) = \frac{p(o^{(t)}|c)p(c|\{o^{(1)}, o^{(2)}, \cdots, o^{(t-1)}\})}{\sum_c^{\mathcal{C}} p(o^{(t)}|c)p(c|\{o^{(1)}, o^{(2)}, \cdots, o^{(t-1)}\})}, \tag{5}$$

To outline a decision theory within this framework, observe that, at decision instant $t$, a Bayesian agent could represent its prior preference for different world possibilities in the form of a probability

distribution over the possible outcomes in $\mathcal{X}$, conditioned on desirability information obtained in earlier decisions, $p(x|c^{(t)}, \{r^{(1)}, r^{(2)}, \cdots r^{(t-1)}\})$. New evidence for the desirability of outcomes observed in context $c^{(t)}$ is incorporated using $p(r^{(t)}|x, c^{(t)})$, a distribution encoding the relative desirability information obtained from the environment at the current time step, conditioned on the context in which the information is obtained. This formulation immediately yields the belief update,

$$p(x|c^{(t)}, r^{(t)}) \propto p(r^{(t)}|c^{(t)}, x) \times p(x|c^{(t)}, \{r^{(1)}, r^{(2)}, \cdots r^{(t-1)}\}), \tag{6}$$

to obtain a posterior probability encoding the desirability of different possibilities $x$, while also accounting tractably for the context in which desirability information is obtained at every decision instance. Defining a choice function to select the mode of the posterior belief completes a rational context-sensitive decision theory.

## 4 Demonstrations

To demonstrate the value of the relative desirability-based encoding of preferences, in Section 4.1, we describe situations in which the influence of context shifting significantly affects human preference behavior in ways that utility-based decision theories have historically been hard-pressed to explain. Complementarily, in Section 4.2 we characterize conditions under which the relative desirability framework yields predictions of choice behavior equivalent to that predicted by ordinal utility theories, and hence, is an equivalent representation for encoding preferences.

### 4.1 Where context matters ...

In this section, we show how our inductive theory of context-sensitive value inference leads, not surprisingly, to a simple explanation for the major varieties of context effects seen in behavioral experiments. These are generally enumerated as attraction, similarity, comparison and reference point effects [2]. Interestingly, we find that each of these effects can be described as a special case of the frog legs example, with the specialization arising out of additional assumptions made about the relationship of the new option added to the choice set. Table 1, with some abuse of notation, describes this relationship between the effects in set-theoretic terms. Space constraints necessitate

| Effect name | Description | Assumptions |
|:---:|:---:|:---:|
| Frog legs | $c_1 \leftarrow \{X, Y\} \Rightarrow X \succ Y, c_2 \leftarrow \{X, Y, Z\} \Rightarrow Y \succ X$ | - |
| Similarity | $c_1 \leftarrow \{X, Y\} \Rightarrow X \succ Y, c_2 \leftarrow \{X, Y, Z\} \Rightarrow Y \succ X$ | $Z \approx X$ |
| Attraction | $c_1 \leftarrow \{X, Y\} \Rightarrow X \sim Y, c_2 \leftarrow \{X, Y, Z\} \Rightarrow X \succ Y$ | $X \succ Z$ |
| Compromise | $c_1 \leftarrow \{X, Y\} \Rightarrow X \succ Y, c_2 \leftarrow \{X, Y, Z\} \Rightarrow Y \succ X$ | $Y \succ^{(c)} X, Z$ |
| Reference point | $c_1 \leftarrow \{X, Y\} \Rightarrow X \sim Y, c_2 \leftarrow \{X, Y, Z\} \Rightarrow X \succ^{(-)} Y$ | $Z \succ X$ |

Table 1: A unified description of context effects. $\succ$ indicates stochastic preference for one item over another. $\succ^{(c)}$ indicates that the preference in question holds only in some observation contexts. $\succ^{(-)}$ indicates that the preference in question is stochastically weaker than before.

an abbreviate description of our results. Detailed descriptions of these effects, supplemented with an explanation of how they may be elicited in our framework, is provided in SI. We use available space to completely describe how the most general version of preference reversal, as seen in the frog legs example, emerges from our framework and provide a brief overview of the other results. To instantiate our likelihood definition in (5), we define a specific mismatch probability,

$$p(\neg y_i^t | z_i^t) = \frac{1}{|\mathcal{X}|} \left( (1 - z_i^t) y_i^t + (1 - y_i^t) z_i^t \right), \tag{7}$$

with $\beta = 1$ for all our demonstrations.

In the frog legs example, the reversal in preferences is anecdotally explained by the diner originally forming a low opinion of the restaurant's chef, given the paucity of choices on the menu, deciding to pick the safe salmon over a possibly a burnt steak. However, the waiter's presenting frog legs as the daily special suddenly raises the diner's opinion of the chef's abilities, causing him to favor steak. This intuition maps very easily into our framework of choice selection, wherein the diner's partial

menu observations $o_1 = \{\text{steak, salmon}\}$ and $o_2 = \{\text{steak, salmon, frog legs}\}$ are associated with two separate contexts $c_1$ and $c_2$ of observing the menu $\mathcal{X}$. Bad experiences related to ordering steak in menus typically observed under context $c_1$ (interpretable as 'cheap restaurants') may be encoded by defining the vector $m = \{1, 0, 0, 0\}$ for $c_1$ and good experiences ordering steak off menues observed in context $c_2$ (interpretable as 'upscale restaurants') as $m = \{0, 1, 0, 0\}$ for $c_2$. Then, by definition, $p(r|\text{salmon}, c_1) > p(r|\text{steak}, c_1)$, while $p(r|\text{salmon}, c_2) < p(r|\text{steak}, c_2)$. For the purposes of this demonstration, let us assume these probability pairs, obtained through the diner's past experiences in restaurants to be $\{0.7, 0.3\}$ and $\{0.3, 0.7\}$ respectively. Now, when the waiter first offers the diner a choice between steak or salmon, the diner computes relative desirabilities using (4), where the only context for the observation is $\{\text{salmon, steak}\}$. Hence, the relative desirabilities of steak and salmon are computed over a single context, and are simply $R(\text{salmon}) = 0.7, R(\text{steak}) = 0.3$. When the diner is next presented with the possibility of ordering frog legs, he now has two possible contexts to evaluate the desirability of his menu options: $\{\text{salmon, steak}\}$ and $\{\text{salmon, steak, frog legs}\}$. Based on the sequence of his history of experience with both contexts, the diner will have some posterior belief $p(c) = \{p, 1 - p\}$ on the two contexts. Then, the relative desirability of salmon, after having observed frog legs on the menu can be calculated using (4) as,

$$R(\text{salmon}) = \frac{p(r|\text{salmon}, c_1)p(\text{salmon}|c_1)p(c_1) + p(r|\text{salmon}, c_2)p(\text{salmon}|c_2)p(c_2)}{p(\text{salmon}|c_1)p(c_1) + p(\text{salmon}|c_2)p(c_2)},$$

$$= \frac{0.7 \times 1 \times p + 0.3 \times 1 \times (1 - p)}{1 \times p + 1 \times (1 - p)} = 0.7p + 0.3(1 - p).$$

Similarly, we obtain $R(\text{steak}) = 0.3p + 0.7(1 - p)$. Clearly, for $1 - p > p$, $R(\text{steak}) > R(\text{salmon})$, and the diner would be rational in switching his preference. Thus, through our inferential machinery, we retrieve the anecdotal explanation for the diner's behavior: if he believes that he is more likely to be in a good restaurant (with probability $(1 - p)$) than not, he will prefer steak.

Along identical lines, making reasonable assumptions about the contexts of past observations, our decision framework accomodates parsimonious explanations for each of the other effects detailed in Table 1. Attraction effects are traditionally studied in market research settings where a consumer is unsure about which of two items to prefer. The introduction of a third item that is clearly inferior to one of the two earlier options leads the consumer towards preferring that particular earlier option. Our framework elicits this behavior through the introduction of additional evidence of the desirability of one of the options from a new context, causing the relative desirability of this particular option to rise. Similarity effects arise when, given that a consumer prefers one item to another, giving him further options that resemble his preferred item causes him to subsequently prefer the item he earlier considered inferior. This effect is elicited simply as a property of division of probability among multiple similar options, resulting in reduced desirabiliy of the previously superior option. Compromise effects arise when the introduction of a third option to a choice set where the consumer already prefers one item to another causes the consumer to consider the previously inferior option as a compromise between the formerly superior option and the new option, and hence prefer it. We find that the compromise effect arises through a combination of reduction in the desirability of the superior option through negative comparions with the new item and increase in the desirability of the formerly inferior item through positive comparisons with the new item, and that this inference occurs automatically in our framework assuming equal history of comparisons between the existing choice set items and the new item. Reference point effects have typically not been associated with explicit studies of context variation, and may in fact be used to reference a number of behavior patterns that do not satisfy the definition we provide in Table 1. Our definition of the reference point effect is particularized to explain data on pain perception collected by [23], demonstrating relativity in evaluation of objectively identical pain conditions depending on the magnitude of alternatively experienced pain conditions. In concord with empirical observation, we show that the relative (un)desirability of an intermediate pain option reduces upon the experience of greater pain, a simple demonstration of prospect relativity that utility-based accounts of value cannot match.

Competing hypotheses that seek to explain these behaviors are either normative and static, (e.g. extended discrete choice models ( [13] provides a recent review), componential context theory [21], quantum cognition [8]) or descriptive and dynamic, (specifically, decision field theory [3]). In contrast, our approach not only takes a dynamic inductive view of value elicitation, it retains a normativity criterion (Bayes rationality) for falsifying observed predictions, a standard that is expected of any *rational* model of decision-making [6].

## 4.2    ... and where it doesn't

It could be conjectured that the relative desirability indicator $d$ will be an inadequate representation of preference information compared with scalar utility signals assigned to each world possibility, which would leave open the possibility that we may have retrieved a context-sensitive decision theory at the expense of theoretical assurance of rational choice selection, as has been the case in many previous attempts cited above. Were this conjecture to be true, it would severely limit the scope and applicability of our proposal. To anticipate this objection, we theoretically prove that our framework reduces to the standard utility-based representation of preferences under equivalent epistemic conditions, showing that our theory retains equivalent rational representational ability as utility theory in simple, and simply extends this representational ability to explain preference behaviors that utility theory can't.

What does it mean for a measure to represent preference information? To show that a utility function $u$ completely represents a preference relation on $\mathcal{X}$ it is sufficient [12] to show that, $\forall x_1, x_2 \in \mathcal{X}, x_1 \succ x_2 \Leftrightarrow u(x_1) > u(x_2)$. Hence, equivalently, to show that our measure of relative desirability $R$ also completely represents preference information, it should be sufficient to show that, for any two possibilities $x_i, x_j \in \mathcal{X}$, and for any observation context $c$

$$x_i \succ x_j \Leftrightarrow R(x_i) > R(x_j). \tag{8}$$

In SI, we prove that (8) holds at decision instant $t$ under three conditions,

(I) **Context consistency:** $\exists c \in \mathcal{C}, s.t. \ x_i \succ x_j \Rightarrow x_i \succ x_j \forall c \in \mathcal{C}_{ij}, \{x_i, x_j\} \in \mathcal{C}_{ij} \subseteq \mathcal{C}.$

(II) **Transitivity between contexts:** if $x_i \succ x_j$ in $c_1$ and $x_j \succ x_k$ in $c_2, \forall c \in \mathcal{C}, x_i \succ x_k$.

(III) **Symmetry in context observability:** $\forall x_i, x_j \in \mathcal{X}, \lim_{t \to \infty} |\mathcal{C}_{i \setminus j}^{(t)}| = |\mathcal{C}_{j \setminus i}^{(t)}|.$[2]

Of the three assumptions we need to prove this equivalence result, **(I)** and **(II)** simply define a stable preference relation across observation contexts and find exact counterparts in the completeness and transitivity assumptions necessary for representing preferences using ordinal utility functions. **(III)**, the only additional assumption we require, ensures that the agent's history of partial observations of the environment does not contain any useful information. The restriction of infinite data observability, while stringent and putatively implausible, actually uncovers an underlying epistemological assumption of utility theory, viz. that utility/desirability values can somehow be obtained *directly* from the environment. Any inference based preference elicitation procedure will therefore necessarily need infinite data to attain formal equivalence with the utility representation. Finally, we point out that our equivalence result does not require us to assume continuity or the equivalent Archimedean property to encode preferences, as required in ordinal utility definitions. This is because the continuity assumption is required as a technical condition in mapping a discrete mathematical object (a preference relation) to a continuous utility function. Since relative desirability is defined constructively on $Q \subseteq \mathbb{Q}, |Q| < \infty$, a continuity assumption is not needed.

## 5    Discussion

Throughout this exegesis, we have encountered three different representations of choice preferences: relative (ordinal) utilities, absolute (cardinal) utilities and our own proposal, viz. relative desirability. Each representation leads to a slightly different definition of rationality, so that, assuming a rational set selection function $\sigma$ in each case we have,

- **Economic rationality:** $x \in \sigma(\mathcal{X}) \Rightarrow \nexists y \in \mathcal{X}, s.t. \ y \succ x$, predominantly used in human preference modeling in neoclassical economics [12]], e.g. discrete choice modeling [9].
- **VNM-rationality:** $x \in \sigma(\mathcal{X}) \Rightarrow \nexists y \in \mathcal{X}, s.t. \ u(y) > u(x)$, predominantly used in studying decision-making under risk [19], e.g. reinforcement learning [1].
- **Bayes rationality:** $x \in \sigma(\mathcal{X}) \Rightarrow \nexists y \in \mathcal{X}, \ s.t. \ R(y, \{H\}) > R(x, \{H\})$, which we have proposed. The term $\{H\}$ here is shorthand for $\{o_1, o_2, \cdots, o_{t-1}\}, \{r_1, r_2, \cdots r_{t-1}\}$, the entire history of choice set and relative desirability observations made by an agent leading up to the current decision instance.

Bayes rationality simply claims that value inference with the same history of partial observations will lead to a consistent preference for a particular option in discrete choice settings. In Section 4.2, we have shown conditions on choice set observations under which Bayes-rationality will be equivalent to economic rationality. VNM-rationality is a further specialization of economic rationality, valid for preference relations that, in addition to being complete, transitive and continuous (as required for economic preferences representable via ordinal utilities) also satisfy an independence of irrelevant attributes (IIA) assumption [16]. Bayes-rationality specializes to economic rationality once we instantiate the underlying intuitions behing the completeness and transitivity assumptions in a context-sensitive preference inference theory. Therefore, rational value inference in the form we propose can formally replace static assumptions about preference orderings in microeconomic models that currently exclusively use ordinal utilities [12]. As such, context-sensitive preference elicitation is immediately useful for the nascent agent-based economic modeling paradigm as well as in dynamic stochastic general equilibrium models of economic behavior. Further work is necessary to develop a context-sensitive equivalent of the IIA assumption, which is necessary for our system to be directly useful in modeling decision-making behaviors under uncertainty. However, even in its current form, our inference model can be used in conjunction with existing 'inverse planning' models of utility elicitation from choice data [17] that infer absolute utilities from choice data using extraneous constraints on the form of the utility function from the environment. In such a synthesis, our model could generate a preference relation sensitive to action set observability, which inverse planning models could use along with additional information from the environment to generate absolute utilities that account for observational biases in the agent's history.

A philosophically astute reader will point out a subtle flaw in our inferential definition of rationality. Namely, while we assume an intuitive notion of partial observability of the world, in practice, our agents compile desirability statistics on the set of all possibilities, irrespective of whether they have ever been observed, a problem that is rooted in an inherent limitation of Bayesian epistemology of being restricted to computing probabilities over a fixed set of hypotheses. How can a desirability representation that assumes that observers maintain probabilistic preferences over all possible states of the world be more epistemologically realistic than one that assumes that observers maintain scalar utility values over the same state space[3]? As a partial response to this criticism, we point out that we do not require an ontic commitment to the computation of joint probability distributions on all $x \in \mathcal{X}$. In practice, it is likely that Bayesian computations are implemented in the brain via sampling schemes that, in hierarchical formulations, allow approximating information of the joint distribution as a set of the most likely marginals (in our case, relative desirability in *typical* observation contexts). Neural implementations of such sampling schemes have been proposed in the recent cognitive science literature [20]. Devising a sampling scheme that matches the intuition of context retrieval from memory to supplement our value-inference scheme presents a promising direction for future research.

Another straightforward extension of our framework would imbue observable world possibilities with *attributes*, resulting in the possibility of deriving a more general definition of contexts as clusters in the space of attributes. Such an extension would result in the possibility of transferring preferences to entirely new possibilities, allowing the set $\mathcal{X}$ to be modified dynamically, which would further address the epistemological criticism above. Even further, such an extension maps directly to the intuition of value inference resulting from organisms' monitoring of internal need states, here modeled as attributes. Canini's recent modeling of transfer learning using hierarchical Dirichlet processes [4] provides most of the mathematical apparatus required to perform such an extension, making this a promising direction for future work in our project.

In conclusion, it has long been recognized that state-specific utility representations of the desirability of options are insufficient to capture the rich variety of systematic behavior patterns that humans exhibit. In this paper, we show that reformulating the atomic unit of desirability as a context-sensitive 'pointer' to the best option in the observed set recovers a rational way of representing desirability in a manner sufficiently powerful to describe a broad range of context effects in decisions. Since it is likely that preferences for options do not exist *a priori* and are induced via experience, our present proposal is expected to approximate the true mechanisms for the emergence of context-sensitive preference variation better than alternative static theories, while retaining normativity criteria missing in alternative dynamic accounts.

## Footnotes

[2]The notation $\mathcal{C}_{i \setminus j}$ references the subset of all observed contexts that contain $x_i$ but not $x_j$.

[3] One could argue that we are essentially observing the state space (to be able to index using its membership), but pretending to not observe it.

# References

[1] A.G. Barto and R.S. Sutton. *Reinforcement Learning: an introduction*. Univesity of Cambridge Press, 1998.

[2] J. R. Busemeyer, R. Barkan, S. Mehta, and A. Chaturvedi. Context effects and models of preferential choice: implications for consumer behavior. *Marketing Theory*, 7(1):39–58, 2007.

[3] J.R. Busemeyer and J.T. Townsend. Decision field theory: A dynamic cognition approach to decision making. *Psychological Review*, 100:432–459, 1993.

[4] K. Canini, M. Shashkov, and T. Griffiths. Modeling transfer learning in human categorization with the hierarchical dirichlet process. In *ICML*, pages 151–158, 2010.

[5] U. Chajewska, D. Koller, and D. Ormoneit. Learning an agent's utility function by observing behavior. In *ICML*, pages 35–42, 2001.

[6] N. Chater. Rational and mechanistic perspectives on reinforcement learning. *Cognition*, 113(3):350 – 364, 2009. Reinforcement learning and higher cognition.

[7] N. Daw and M. Frank. Reinforcement learning and higher level cognition: Introduction to special issue. *Cognition*, 113(3):259 – 261, 2009. Reinforcement learning and higher cognition.

[8] L. Gabora and D. Aerts. Contextualizing concepts using a mathematical generalization of the quantum formalism. *Joural of Experimental and Theoretical Artificial Intelligence*, 14(4):327–358, 2002.

[9] D. Hensher, J. Rose, and W. Greene. *Applied Choice Analysis: A Primer*. Cambridge University Press, 2005.

[10] A. Jern, C. Lucas, and C. Kemp. Evaluating the inverse decision-making approach to preference learning. In *NIPS*, pages 2276–2284, 2011.

[11] D. Kahneman. Perception, action and utility: the tangled skein. In M. Rabinovich, K. Friston, and P. Varona, editors, *Principles of Brain Dynamics: Global State Interactions*. MIT Pres, 2012.

[12] D. Kreps. *A Course in Microeconomic Theory*, pages 17–69. Princeton University Press, 1990.

[13] W. Leong and D. Hensher. Embedding decision heuristics in discrete choice models: A review. *Transport Reviews*, 32(3):313–331, 2012.

[14] C.G. Lucas, T. Griffiths, F. Xu, and C. Fawcett. A rational model of preference learning and choice prediction by children. In *NIPS*, pages 985–992, 2008.

[15] R. D. Luce and H. Raiffa. *Games and Decisions: Introduction and Critical Survey*. Wiley, New York, 1957.

[16] J.v. Neumann and O. Morgenstern. *Theory of Games and Economic Behavior*. Princeton University Press, 1953.

[17] A. Y. Ng and S. J. Russell. Algorithms for inverse reinforcement learning. In *Proceedings of the Seventeenth International Conference on Machine Learning*, ICML '00, pages 663–670, 2000.

[18] M. Rabin. Psychology and economics. *Journal of Economic Literature*, 36(1):pp. 11–46, 1998.

[19] S.J. Russell and P. Norvig. *Artificial Intelligence: A Modern Approach*. MIT Press, 1998.

[20] L. Shi and T. Griffiths. Neural Implementation of Hierarchical Bayesian Inference by Importance Sampling. In Y. Bengio, D. Schuurmans, J. Lafferty, C. K. I. Williams, and A. Culotta, editors, *Advances in Neural Information Processing Systems 22*, pages 1669–1677. 2009.

[21] A. Tversky and I. Simonson. Context-dependent preferences. *Management Science*, 39(10):pp. 1179–1189, 1993.

[22] I. Vlaev, N. Chater, N. Stewart, and G. Brown. Does the brain calculate value? *Trends in Cognitive Sciences*, 15(11):546 – 554, 2011.

[23] I. Vlaev, B. Seymour, R.J. Dolan, and N. Chater. The price of pain and the value of suffering. *Psychological Science*, 20(3):309–317, 2009.

